# Phase-Space Learning

**Fu-Sheng Tsung**
Chung Tai Ch'an Temple
56, Yuon-fon Road, Yi-hsin Li, Pu-li
Nan-tou County, Taiwan 545
Republic of China

**Garrison W. Cottrell***
Institute for Neural Computation
Computer Science & Engineering
University of California, San Diego
La Jolla, California 92093

## Abstract

Existing recurrent net learning algorithms are inadequate. We introduce the conceptual framework of viewing recurrent training as matching vector fields of dynamical systems in phase space. Phase-space reconstruction techniques make the hidden states explicit, reducing temporal learning to a feed–forward problem. In short, we propose viewing *iterated prediction* [LF88] as the best way of training recurrent networks on deterministic signals. Using this framework, we can train multiple trajectories, insure their stability, and *design* arbitrary dynamical systems.

## 1 INTRODUCTION

Existing general-purpose recurrent algorithms are capable of rich dynamical behavior. Unfortunately, straightforward applications of these algorithms to training fully–recurrent networks on complex temporal tasks have had much less success than their feedforward counterparts. For example, to train a recurrent network to oscillate like a sine wave (the "hydrogen atom" of recurrent learning), existing techniques such as Real Time Recurrent Learning (RTRL) [WZ89] perform suboptimally. Williams & Zipser trained a two-unit network with RTRL, with one teacher signal. One unit of the resulting network showed a distorted waveform, the other only half the desired amplitude. [Pea89] needed four hidden units. However, our work demonstrates that a two-unit recurrent network with no hidden units can learn the sine wave very well [Tsu94]. Existing methods also have several other

limitations. For example, networks often fail to converge even though a solution is known to exist; teacher forcing is usually necessary to learn periodic signals; it is not clear how to train multiple trajectories at once, or how to insure that the trained trajectory is stable (an *attractor*).

In this paper, we briefly analyze the algorithms to discover why they have such difficulties, and propose a general solution to the problem. Our solution is based on the simple idea of using the techniques of time series prediction as a methodology for recurrent network training.

First, by way of introducing the appropriate concepts, consider a system of coupled autonomous[1] first order network equations:

$$
\begin{aligned}
dx_1/dt &= F_1(x_1(t), x_2(t), \cdots, x_n(t)) \\
dx_2/dt &= F_2(x_1(t), x_2(t), \cdots, x_n(t)) \\
&\vdots \\
dx_n/dt &= F_n(x_1(t), x_2(t), \cdots, x_n(t))
\end{aligned}
$$

or, in vector notation,

$$
X(t) = F(X) \quad \text{where} \quad X(t) = (x_1(t), x_2(t), \cdots, x_n(t))
$$

The **phase space** is the space of the dependent variables (X), it does not include $t$, while the **state space** incorporates $t$. The evolution of a trajectory $X(t)$ traces out a phase curve, or *orbit*, in the $n$-dimensional phase space of $X$. For low dimensional systems (2- or 3-D), it is easy to visualize the limit sets in the phase space: a fixed point and a limit cycle become a single point and a closed orbit (closed curve), respectively. In the state space they become an infinite straight line and a spiral. $F(X)$ defines the **vector field** of $X$, because it associates a vector with each point in the phase space of $X$ whose direction and magnitude determines the movement of that point in the next instant of time (by definition, the tangent vector).

## 2   ANALYSIS OF CURRENT APPROACHES

To get a better understanding of why recurrent algorithms have not been very effective, we look at what happens during training with two popular recurrent learning techniques: RTRL and back propagation through time (BPTT). With each, we illustrate a different problem, although the problems apply equally to each technique.

RTRL is a forward-gradient algorithm that keeps a matrix of partial derivatives of the network activation values with respect to every weight. To train a periodic trajectory, it is necessary to *teacher-force* the visible units [WZ89], i.e., on every iteration, after the gradient has been calculated, the activations of the visible units are replaced by the teacher. To see why, consider learning a pair of sine waves offset by 90°. In phase space, this becomes a circle (Figure 1a). Initially the network

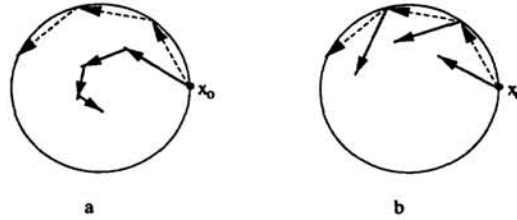

Figure 1: Learning a pair of sine waves with RTRL learning. (a) without teacher forcing, the network dynamics (solid arrows) take it far from where the teacher (dotted arrows) assumes it is, so the gradient is incorrect. (b) With teacher forcing, the network's visible units are returned to the trajectory.

(thick arrows) is at position $X_0$ and has arbitrary dynamics. After a few iterations, it wanders far away from where the teacher (dashed arrows) assumes it to be. The teacher then provides an incorrect next target from the network's current position. Teacher-forcing (Figure 1b), resets the network back on the circle, where the teacher again provides useful information.

However, if the network has hidden units, then the phase space of the visible units is just a projection of the actual phase space of the network, and the teaching signal gives no information as to where the hidden units should be in this higher-dimensional phase space. Hence the hidden unit states, unaltered by teacher forcing, may be entirely unrelated to what they should be. This leads to the *moving targets* problem. During training, every time the visible units re–visit a point, the hidden unit activations will differ, Thus the mapping changes during learning. (See [Pin88, WZ89] for other discussions of teacher forcing.)

With BPTT, the network is unrolled in time (Figure 2). This unrolling reveals another problem: Suppose in the teaching signal, the visible units' next state is a non-linearly separable function of their current state. Then hidden units are needed *between* the visible unit layers, but there are no intermediate hidden units in the unrolled network. The network must thus take *two time steps* to get to the hidden units and back. One can deal with this by giving the teaching signal *every other iteration*, but clearly, this is not optimal (consider that the hidden units must "bide their time" on the alternate steps).[2]

The trajectories trained by RTRL and BPTT tend to be stable in simulations of simple tasks [Pea89, TCS90], but this stability is paradoxical. Using teacher forcing, the networks are trained to go from a point on the trajectory, to a point within the ball defined by the error criterion $\epsilon$ (see Figure 4 (a)). However, after learning, the networks behave such that from a place near the trajectory, they head for the trajectory (Figure 4 (b)). Hence the paradox. Possible reasons are: 1) the hidden unit moving targets provide training *off* the desired trajectory, so that *if* the training is successful, the desired trajectory is stable; 2) we would never consider the training successful if the network "learns" an unstable trajectory; 3) the stable dynamics in typical situations have simpler equations than the unstable dynamics [Nak93]. To create an unstable periodic trajectory would imply the existence of stable regions both inside and outside the unstable trajectory; dynamically this is

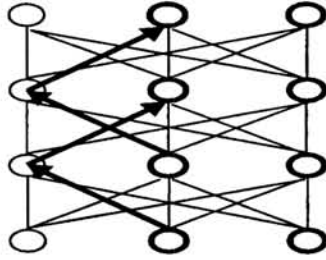

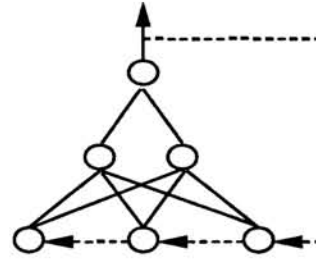

Figure 2: A nonlinearly separable mapping must be computed by the hidden units (the leftmost unit here) every other time step.

Figure 3: The network used for iterated prediction training. Dashed connections are used after learning.

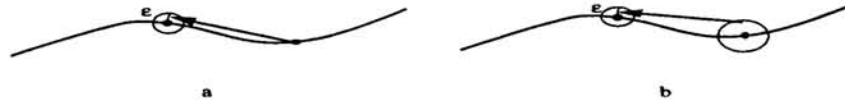

Figure 4: The paradox of attractor learning with teacher forcing. (a) During learning, the network learns to move from the trajectory to a point near the trajectory. (b) After learning, the network moves from nearby points towards the trajectory.

more complicated than a simple periodic attractor. In dynamically complex tasks, a stable trajectory may no longer be the simplest solution, and stability could be a problem.

In summary, we have pointed out several problems in the RTRL (forward-gradient) and BPTT (backward-gradient) classes of training algorithms:

1. Teacher forcing with hidden units is at best an approximation, and leads to the moving targets problem.

2. Hidden units are not placed properly for some tasks.

3. Stability is paradoxical.

## 3   PHASE-SPACE LEARNING

The inspiration for our approach is prediction training [LF88], which at first appears similar to BPTT, but is subtly different. In the standard scheme, a feedforward network is given a time window, a set of previous points on the trajectory to be learned, as inputs. The output is the next point on the trajectory. Then, the inputs are shifted left and the network is trained on the next point (see Figure 3). Once the network has learned, it can be treated as recurrent by iterating on its own predictions.

The prediction network differs from BPTT in two important ways. First, the visible units encode a selected temporal history of the trajectory (the time window). The point of this *delay space embedding* is to reconstruct the phase space of the underlying system. [Tak81] has shown that this can always be done for a deterministic system. Note that in the reconstructed phase space, the mapping from one

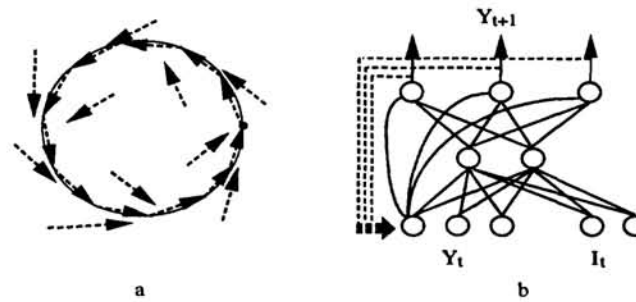

Figure 5: Phase-space learning. (a) The training set is a sample of the vector field. (b) Phase-space learning network. Dashed connections are used after learning.

point to the next (based on the vector field) is deterministic. *Hence what originally appeared to be a recurrent network problem can be converted into an entirely feed forward problem.* Essentially, the delay-space reconstruction makes hidden states visible, and recurrent hidden units unnecessary. Crucially, dynamicists have developed excellent reconstruction algorithms that not only automate the choices of delay and embedding dimension but also filter out noise or get a good reconstruction despite noise [FS91, Dav92, KBA92]. On the other hand, we clearly cannot deal with non-deterministic systems by this method.

The second difference from BPTT is that the hidden units are *between* the visible units, allowing the network to produce nonlinearly separable transformations of the visible units in a single iteration. In the recurrent network produced by iterated prediction, the sandwiched hidden units can be considered "fast" units with delays on the input/output links summing to 1.

Since we are now learning a mapping in phase space, stability is easily ensured by adding additional training examples that converge towards the desired orbit.[3] We can also explicitly *control* convergence speed by the size and direction of the vectors.

Thus, *phase-space learning* (Figure 5) consists of: (1) embedding the temporal signal to recover its phase space structure, (2) generating local approximations of the vector field near the desired trajectory, and (3) functional approximation of the vector field with a feedforward network. Existing methods developed for these three problems can be directly and independently applied to solve the problem. Since feedforward networks are universal approximators [HSW89], we are assured that at least locally, the trajectory can be represented. The trajectory is recovered from the iterated output of the pre–embedded portion of the visible units. Additionally, we may also extend the phase-space learning framework to also include inputs that affect the output of the system (see [Tsu94] for details).[4]

In this framework, training multiple attractors is just training orbits in different parts of the phase space, so they simply add more patterns to the training set. In fact, we can now create *designer dynamical systems* possessing the properties we want, e.g., with combinations of fixed point, periodic, or chaotic attractors.

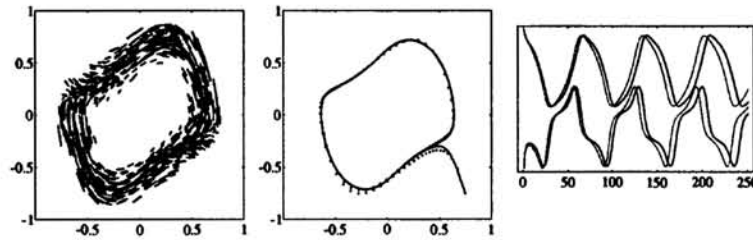

Figure 6: Learning the van der Pol oscillator. (a) the training set. (b) Phase space plot of network (solid curve) and teacher (dots). (c) State space plot.

As an example, to store any number of arbitrary periodic attractors $z_i(t)$ with periods $T_i$ in a single recurrent network, create two new coordinates for each $z_i(t)$, $(x_i(t), y_i(t)) = (sin(\frac{2\pi}{T_i}t), cos(\frac{2\pi}{T_i}t))$, where $(x_i, y_i)$ and $(x_j, y_j)$ are disjoint circles for $i \neq j$. Then $(x_i, y_i, z_i)$ is a valid embedding of all the periodic attractors in phase space, and the network can be trained. In essence, the first two dimensions form "clocks" for their associated trajectories.

## 4   SIMULATION RESULTS

In this section we illustrate the method by learning the van der Pol oscillator (a much more difficult problem than learning sine waves), learning four separate periodic attractors, and learning an attractor inside the basin of another attractor.

### 4.1   LEARNING THE VAN DER POL OSCILLATOR

The van der Pol equation is defined by:

$$\dot{x} = y \qquad \dot{y} = -\alpha(x^2 - b)y - \omega^2 x$$

We used the values 0.7, 1, 1 for the parameters $\alpha$, $b$, and $\omega$, for which there is a global periodic attractor (Figure 6). We used a step size of 0.1, which discretizes the trajectory into 70 points. The network therefore has two visible units. We used two hidden layers with 20 units each, so that the unrolled, feedforward network has a 2-20-20-2 architecture. We generated 1500 training pairs using the vector field near the attractor. The learning rate was 0.01, scaled by the fan-in, momentum was 0.75, we trained for 15000 epochs. The order of the training pairs was randomized. The attractor learned by the network is shown in (Figure 6 (b)). Comparison of the temporal trajectories is shown in Figure 6 (c); there is a slight frequency difference. The average MSE is 0.000136. Results from a network with two layers of 5 hidden units each with 500 data pairs were similar (MSE=0.00034).

### 4.2   LEARNING MULTIPLE PERIODIC ATTRACTORS

[Hop82] showed how to store multiple fixed-point attractors in a recurrent net. [Bai91] can store periodic and chaotic attractors by inverting the normal forms of these attractors into higher order recurrent networks. However, traditional recurrent training offers no obvious method of training multiple attractors. [DY89] were able

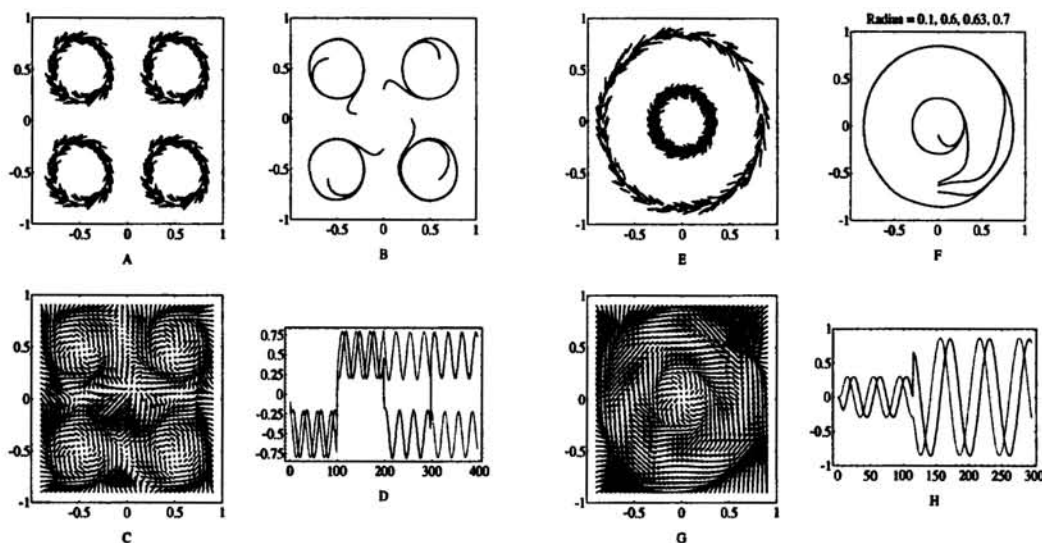

Figure 7: Learning multiple attractors. In each case, a 2-20-20-2 network using conjugate gradient is used. Learning 4 attractors: (A) Training set. (B) Eight trajectories of the trained network. (C) Induced vector field of the network. There are five unstable fixed points. (D) State space behavior as the network is "bumped" between attractors. Learning 2 attractors, one inside the other: (E) Training set. (F) Four trajectories of the trained network. (G) Induced vector field of the network. There is an unstable limit cycle between the two stable ones. (H) State space behavior with a "bump".

to store two limit cycles by starting with fixed points stored in a Hopfield net, and training each fixed point locally to become a periodic attractor. Our approach has no difficulty with multiple attractors. Figure 7 (A-D) shows the result of training four coexisting periodic attractors, one in each quadrant of the two-dimensional phase space. The network will remain in one of the attractor basins until an external force pushes it into another attractor basin. Figure 7 (E-H) shows a network with two periodic attractors, this time one inside the other. This vector field possess an unstable limit cycle between the two stable limit cycles. This is a more difficult task, requiring 40 hidden units, whereas 20 suffice for the previous task (not shown).

## 5   SUMMARY

We have presented a phase space view of the learning process in recurrent nets. This perspective has helped us to understand and overcome some of the problems of using traditional recurrent methods for learning periodic and chaotic attractors. Our method can learn multiple trajectories, explicitly insure their stability, and avoid overfitting; in short, we provide a practical approach to learning complicated temporal behaviors. The phase-space framework essentially breaks the problem into three sub-problems: (1) Embedding a temporal signal to recover its phase space structure, (2) generating local approximations of the vector field near the desired trajectory, and (3) functional approximation in feedforward networks. We have demonstrated that using this method, networks can learn complex oscillations and multiple periodic attractors.

**Acknowledgements**

This work was supported by NIH grant R01 MH46899-01A3. Thanks for comments from Steve Biafore, Kenji Doya, Peter Rowat, Bill Hart, and especially Dave DeMers for his timely assistance with simulations.

## Footnotes

*Correspondence should be addressed to the second author: gary@cs.ucsd.edu

[1]Autonomous means the right hand side of a differential equation does not explicitly reference $t$, e.g. $dx/dt = 2x$ is autonomous, even though $x$ is a function of $t$, but $dx/dt = 2x+t$ is not. Continuous neural networks without inputs are autonomous. A nonautonomous system can always be turned into an autonomous system in a higher dimension.

[2]At NIPS, 0 delay connections to the hidden units were suggested, which is essentially part of the solution given here.

[3] The common practice of adding noise to the input in prediction training is just a simple minded way of adding convergence information.

[4] Principe & Kuo(this volume) show that for chaotic attractors, it is better to treat this as a recurrent net and train using the predictions.

# References

[Bai91]  W. Baird and F. Eeckman. Cam storage of analog patterns and continuous sequences with $3n^2$ weights. In R.P. Lippmann, J.E. Moody, and D.S. Touretzky, editors, *Advances in Neural Information Processing Systems*, volume 3, pages 91–97, 1991. Morgan Kaufmann, San Mateo.

[Dav92]  M. Davies. Noise reduction by gradient descent. *International Journal of Bifurcation and Chaos*, 3:113–118, 1992.

[DY89]  K. Doya and S. Yoshizawa. Memorizing oscillatory patterns in the analog neuron network. In *IJCNN*, Washington D.C., 1989. IEEE.

[FS91]  J.D. Farmer and J.J. Sidorowich. Optimal shadowing and noise reduction. *Physica D*, 47:373–392, 1991.

[Hop82]  J.J. Hopfield. Neural networks and physical systems with emergent collective computational abilities. *Proceedings of the National Academy of Sciences, USA*, 79, 1982.

[HSW89]  K. Hornik, M. Stinchcombe, and H. White. Multilayer feedforward networks are universal approximators. *Neural Networks*, 2:359–366, 1989.

[KBA92]  M.B. Kennel, R. Brown, and H. Abarbanel. Determining embedding dimension for phase-space reconstruction using a geometrical construction. *Physical Review A*, 45:3403–3411, 1992.

[LF88]  A. Lapedes and R. Farber. How neural nets work. In D.Z. Anderson, editor, *Neural Information Processing Systems*, pages 442–456, Denver 1987, 1988. American Institute of Physics, New York.

[Nak93]  Hiroyuki Nakajima. A paradox in learning trajectories in neural networks. Working paper, Dept. of EE II, Kyoto U., Kyoto, JAPAN, 1993.

[Pea89]  B.A. Pearlmutter. Learning state space trajectories in recurrent neural networks. *Neural Computation*, 1:263–269, 1989.

[Pin88]  F.J. Pineda. Dynamics and architecture for neural computation. *Journal of Complexity*, 4:216–245, 1988.

[Tak81]  F. Takens. Detecting strange attractors in turbulence. In D.A. Rand and L.-S. Young, editors, *Dynamical Systems and Turbulence*, volume 898 of *Lecture Notes in Mathematics*, pages 366–381, Warwick 1980, 1981. Springer-Verlag, Berlin.

[TCS90]  F-S. Tsung, G. W. Cottrell, and A. I. Selverston. Some experiments on learning stable network oscillations. In *IJCNN*, San Diego, 1990. IEEE.

[Tsu94]  F-S. Tsung. *Modelling Dynamical Systems with Recurrent Neural Networks*. PhD thesis, University of California, San Diego, 1994.

[WZ89]  R.J. Williams and D. Zipser. A learning algorithm for continually running fully recurrent neural networks. *Neural Computation*, 1:270–280, 1989.